# Characterizing Neurons in the Primary Auditory Cortex of the Awake Primate Using Reverse Correlation

**R. Christopher deCharms**
decharms@phy.ucsf.edu

**Michael M. Merzenich**
merz@phy.ucsf.edu

W. M. Keck Center for Integrative Neuroscience
University of California, San Francisco CA 94143

## Abstract

While the understanding of the functional role of different classes of neurons in the awake primary visual cortex has been extensively studied since the time of Hubel and Wiesel (Hubel and Wiesel, 1962), our understanding of the feature selectivity and functional role of neurons in the primary auditory cortex is much farther from complete. Moving bars have long been recognized as an optimal stimulus for many visual cortical neurons, and this finding has recently been confirmed and extended in detail using reverse correlation methods (Jones and Palmer, 1987; Reid and Alonso, 1995; Reid et al., 1991; Ringach et al., 1997). In this study, we recorded from neurons in the primary auditory cortex of the awake primate, and used a novel reverse correlation technique to compute receptive fields (or preferred stimuli), encompassing both multiple frequency components and ongoing time. These spectrotemporal receptive fields make clear that neurons in the primary auditory cortex, as in the primary visual cortex, typically show considerable structure in their feature processing properties, often including multiple excitatory and inhibitory regions in their receptive fields. These neurons can be sensitive to stimulus edges in frequency composition or in time, and sensitive to stimulus transitions such as changes in frequency. These neurons also show strong responses and selectivity to continuous frequency modulated stimuli analogous to visual drifting gratings.

## 1 Introduction

It is known that auditory neurons are tuned for a number of independent feature parameters of simple stimuli including frequency (Merzenich et al., 1973), intensity (Sutter and Schreiner, 1995), amplitude modulation (Schreiner and Urbas, 1988), and

others. In addition, auditory cortical responses to multiple stimuli can enhance or suppress one another in a time dependent fashion (Brosch and Schreiner, 1997; Phillips and Cynader, 1985; Shamma and Symmes, 1985), and auditory cortical neurons can be highly selective for species-specific vocalizations (Wang et al., 1995; Wollberg and Newman, 1972), suggesting complex acoustic processing by these cells. It is not yet known if these many independent selectivities of auditory cortical neurons reflect a discernible underlying pattern of feature decomposition, as has often been suggested (Merzenich et al., 1985; Schreiner and Mendelson, 1990; Wang et al., 1995). Further, since sustained firing rate responses in the auditory cortex to tonal stimuli are typically much lower than visual responses to drifting bars (deCharms and Merzenich, 1996b), it has been suggested that the preferred type of auditory stimulus may still not be known (Nelken et al., 1994). We sought to develop an unbiased method for determining the full feature selectivity of auditory cortical neurons, whatever it might be, in frequency and time based upon reverse correlation.

## 2   Methods

Recordings were made from a chronic array of up to 49 individually placed ultra-fine extracellular Iridium microelectrodes, placed in the primary auditory cortex of the adult owl monkey. The electrodes had tip lengths of 10-25microns, which yield impedance values of .5-5MOhm and good isolation of signals from individual neurons or clusters of nearby neurons. We electrochemically activated these tips to add an ultramicroscopic coating of Iridium Oxide, which leaves the tip geometry unchanged, but decreases the tip impedance by more than an order of magnitude, resulting in substantially improved recording signals. These signals are filtered from .3-8kHz, sampled at 20kHz, digitized, and sorted. The stimuli used were a variant of random

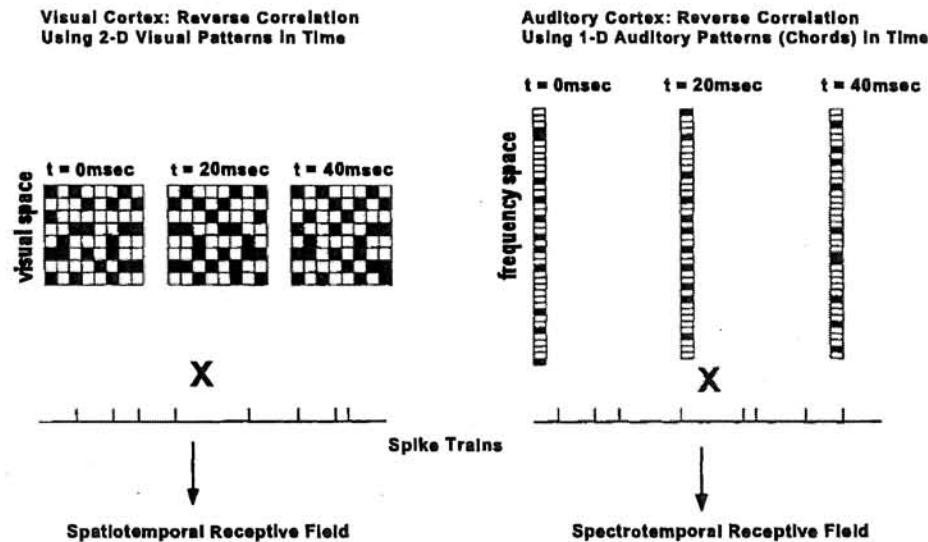

Figure 1: Schematic of stimuli used for reverse correlation.

white noise which was designed to allow us to characterize the responses of neurons in time and in frequency. As shown in figure 1, these stimuli are directly analogous to stimuli that have been used previously to characterize the response properties of neurons in the primary visual cortex (Jones and Palmer, 1987; Reid and Alonso, 1995; Reid et al., 1991). In the visual case, stimuli consist of spatial checkerboards that span some portion of the two-dimensional visual field and change pattern with a short sampling interval. In the auditory case, which we have studied here, the stimuli chosen were randomly selected chords, which approximately evenly span a

portion of the one-dimensional receptor surface of the cochlea. These stimuli consist of combinations of pure tones, all with identical phase and all with 5 msec cosine-shaped ramps in amplitude when they individually turn on or off. Each chord was created by randomly selecting frequency values from 84 possible values which span 7 octaves from 110Hz to 14080Hz in even semitone steps. The density of tones in each stimulus was 1 tone per octave on average, or 7 tones per chord, but the stimuli were selected stochastically so a given chord could be composed of a variable number of tones of randomly selected frequencies. We have used sampling rates of 10-100 chords/second, and the data here are from stimuli with 50 chords/second. Stimuli with random, asynchronous onset times of each tone produce similar results. These stimuli were presented in the open sound field within an acoustical isolation chamber at 44.1kHz sampling rate directly from audio compact disk, while the animal sat passively in the sound field or actively performed an auditory discrimination task, receiving occasional juice rewards. The complete characterization set lasted for ten minutes, thereby including 30,000 individual chords.

Spike trains were collected from multiple sites in the cortex simultaneously during the presentation of our characterization stimulus set, and individually reverse correlated with the times of onset of each of the tonal stimuli. The reverse correlation method computes the number of spikes from a neuron that were detected, on average, during a given time preceding, during, or following a particular tonal stimulus component from our set of chords. These values are presented in spikes/s for all of the tones in the stimulus set, and for some range of time shifts. This method is somewhat analogous in intention to a method developed earlier for deriving spectrotemporal receptive fields for auditory midbrain neurons (Eggermont et al., 1983), but previous methods have not been effective in the auditory cortex.

# 3   Results

Figure 2 shows the spectrotemporal responses of neurons from four locations in the primary auditory cortex. In each panel, the time in milliseconds between the onset of a particular stimulus component and a neuronal spike is shown along the horizontal axis. Progressively greater negative time shifts indicate progressively longer latencies from the onset of a stimulus component until the neuronal spikes. The frequency of the stimulus component is shown along the vertical axis, in octave spacing from a 110Hz standard, with twelve steps per octave. The brightness corresponds to the average rate of the neuron, in spk/s, driven by a particular stimulus component. The reverse-correlogram is thus presented as a stimulus triggered spike rate average, analogous to a standard peristimulus time histogram but reversed in time, and is identical to the spectrogram of the estimated optimal stimulus for the cell (a spike triggered stimulus average which would be in units of mean stimulus density).

A minority of neurons in the primary auditory cortex have spectrotemporal receptive fields that show only a single region of increased rate, which corresponds to the traditional characteristic frequency of the neuron, and no inhibitory region. We have found that cells of this type (less than 10%, not shown) are less common than cells with multimodal receptive field structure. More commonly, neurons have regions of both increased and decreased firing rate relative to their mean rate within their receptive fields. For terminological convenience, these will be referred to as excitatory and inhibitory regions, though these changes in rate are not diagnostic of an underlying mechanism. Neurons with receptive fields of this type can serve as detectors of stimulus edges in both frequency space, and in time. The neuron shown in figure 2a has a receptive field structure indicative of lateral inhibition in frequency space. This cell prefers a very narrow range of frequencies, and decreases its firing rate for nearby frequencies, giving the characteristic of a sharply-tuned bandpass filter. This

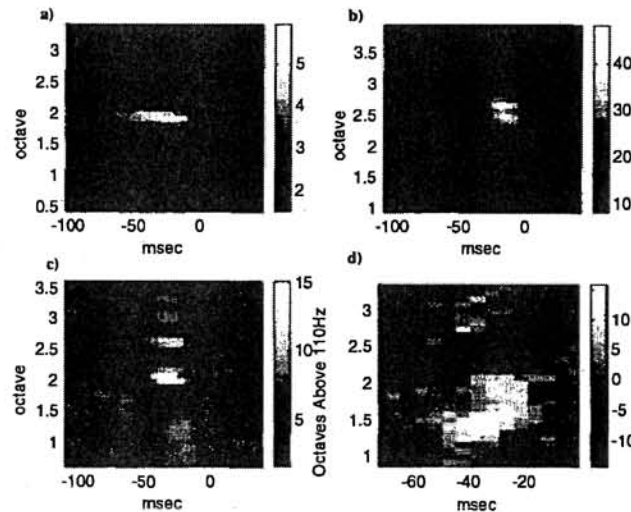

Figure 2: Spectrotemporal receptive fields of neurons in the primary auditory cortex of the awake primate. These receptive fields are computed as described in methods. Receptive field structures read from left to right correspond to a preferred stimulus for the neuron, with light shading indicating more probable stimulus components to evoke a spike, and dark shading indicating less probable components. Receptive fields read from right to left indicate the response of the neuron in time to a particular stimulus component. The colorbars correspond to the average firing rates of the neurons in Hz at a given time preceding, during, or following a particular stimulus component.

type of response is the auditory analog of a visual or tactile edge detector with lateral inhibition. Simple cells in the primary visual cortex typically show similar patterns of center excitation along a short linear segment, surrounded by inhibition (Jones and Palmer, 1987; Reid and Alonso, 1995; Reid et al., 1991). The neuron shown in figure 2b shows a decrease in firing rate caused by a stimulus frequency which at a later time causes an increase in rate. This receptive field structure is ideally suited to detect stimulus transients, and can be thought of as a detector of temporal edges. Neurons in the auditory cortex typically prefer this type of stimulus, which is initially soft or silent and later loud. This corresponds to a neuronal response which shows an increase followed by a decrease in firing rate. This is again analogous to neuronal responses in the primary visual cortex, which also typically show a firing rate pattern to an optimal stimulus of excitation followed by inhibition, and preference for stimulus transients such as when a stimulus is first off and then comes on.

The neuron shown in figures 2c shows an example which has complex receptive field structure, with multiple regions. Cells of this type would be indicative of selectivity for feature conjunctions or quite complex stimuli, perhaps related to sounds in the animal's learned environment. Cells with complex receptive field structures are common in the awake auditory cortex, and we are in the process of quantifying the percentages of cells that fit within these different categories.

Neurons were observed which respond with increased rate to one frequency range at one time, and a different frequency range at a later time, indicative of selectivity for frequency modulations(Suga, 1965). Regions of decreased firing rate can show similar patterns. The neuron shown in figure 2d is an example of this type. This pattern is strongly analogous to motion energy detectors in the visual system (Adelson and Bergen, 1985), which detect stimuli moving in space, and these cells are selective for changes in frequency.

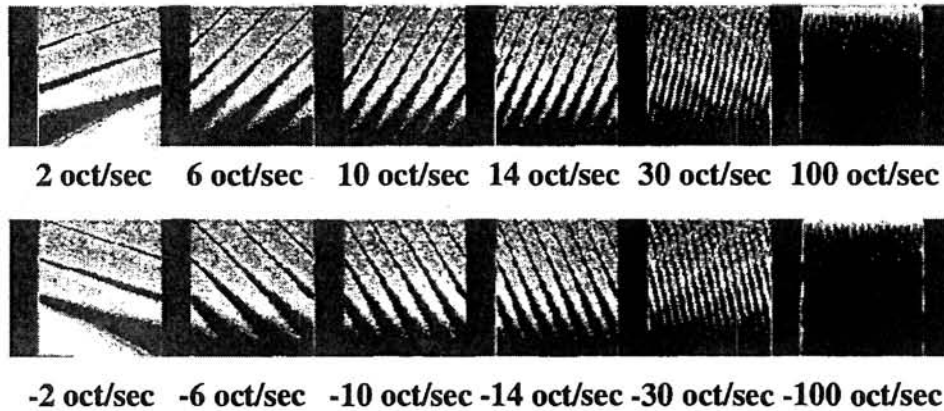

Figure 3: Parametric stimulus set used to explore neuronal responses to continuously changing stimulus frequency. Images are spectrograms of stimuli from left to right in time, and spanning seven octaves of frequency from bottom to top. Each stimulus is one second. Numbers indicate the sweep rate of the stimuli in octaves per second.

Based on the responses shown, we wondered whether we could find a more optimal class of stimuli for these neuron, analogous to the use of drifting bars or gratings in the primary visual cortex. We have created auditory stimuli which correspond exactly to the preferred stimulus computed for a particular cell from the cell's spectrotemporal receptive field (manuscript in preparation), and we have also designed a parametric class of stimuli which are designed to be particularly effective for neurons selective for stimuli of changing amplitude or frequency, which are presented here. The stimuli shown in figure 3 are auditory analogous of visual drifting grating stimuli. The stimuli are shown as spectrograms, where time is along the horizontal axis, frequency content on an octave scale is along the vertical axis, and brightness corresponds to the intensity of the signal. These stimuli contain frequencies that change in time along an octave frequency scale so that they repeatedly pass approximately linearly through a neurons receptive field, just as a drifting grating would pass repeatedly through the receptive field of a visual neuron. These stimuli are somewhat analogous to drifting ripple stimuli which have recently been used by Kowalski, et.al. to characterize the linearity of responses of neurons in the anesthetized ferret auditory cortex (Kowalski et al., 1996a; Kowalski et al., 1996b).

Neurons in the auditory cortex typically respond to tonal stimuli with a brisk onset response at the stimulus transient, but show sustained rates that are far smaller than found in the visual or somatosensory systems (deCharms and Merzenich, 1996a). We have found neurons in the awake animal that respond with high firing rates and significant selectivity to the class of moving stimuli shown in figure 3. An outstanding example of this is shown in figure 4. The neuron in this example showed a very high sustained firing rate to the optimal drifting stimulus, as high as 60 Hz for one second. The neuron shown in this example also showed considerable selectivity for stimulus velocity, as well as some selectivity for stimulus direction.

## 4    Conclusions

These stimuli enable us to efficiently quantify the response characteristics of neurons in the awake primary auditory cortex, as well as producing optimal stimuli for particular neurons. The data that we have gathered thus far extend our knowledge about the complex receptive field structure of cells in the primary auditory cortex,

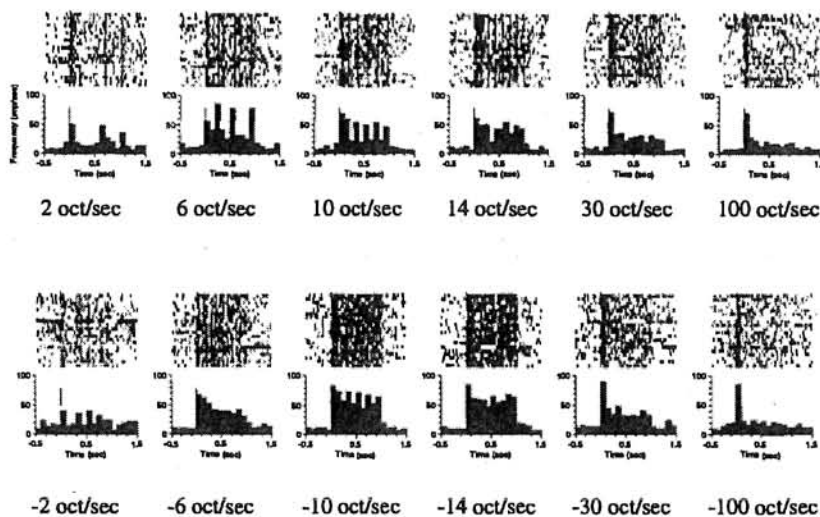

Figure 4: Responses of a neuron in the primary auditory cortex of the awake primate to example stimuli take form our characterization set, as shown in figure 3. In each panel, the average response rate histogram in spikes per second is shown below rastergrams showing the individual action potentials elicited on each of twenty trials.

and show some considerable analogy with neurons in the primary visual cortex. In addition, they indicate that it is possible to drive auditory cortical cells to high rates of sustained firing, as in the visual cortex. This method will allow a number of future questions to be addressed. Since we have recorded many neurons simultaneously, we are interested in the interactions among large populations of neurons and how these relate to stimuli. We are also recording responses to these stimuli while monkeys are performing cognitive tasks involving attention and learning, and we hope that this will give us insight into the effects on cell selectivity of the context provided by other stimuli, the animal's behavioral state or awareness of the stimuli, and the animal's prior learning of stimulus sets.

# 5   References

Adelson EH, Bergen JR (1985) Spatiotemporal energy models for the perception of motion. J. Opt. Soc. Am. A, 2, 284-299.

Brosch M, Schreiner CE (1997) Time course of forward masking tuning curves in cat primary auditory cortex. J Neurophysiol, 77, 923-43.

deCharms RC, Merzenich MM (1996a) Primary cortical representation of sounds by the coordination of action-potential timing. Nature, 381, 610-3.

deCharms RC, Merzenich MM (1996b) Primary cortical representation of sounds by the coordination of action-potential timing. Nature, 381, 610-613.

Eggermont JJ, Aertsen AM, Johannesma PI (1983) Quantitative characterisation procedure for auditory neurons based on the spectro-temporal receptive field. Hear Res, 10, 167-90.

Hubel DH, Wiesel TN (1962) Receptive fields, binocular interaction and functional archtecture in the cat's visual cortex. J. Physiol., 160, 106-154.

Jones JP, Palmer LA (1987) The two-dimensional spatial structure of simple receptive

fields in cat striate cortex. J Neurophysiol, 58, 1187-211.

Kowalski N, Depireux DA, Shamma SA (1996a) Analysis of dynamic spectra in ferret primary auditory cortex. I. Characteristics of single-unit responses to moving ripple spectra. J Neurophysiol, 76, 3503-23.

Kowalski N, Depireux DA, Shamma SA (1996b) Analysis of dynamic spectra in ferret primary auditory cortex. II. Prediction of unit responses to arbitrary dynamic spectra. J Neurophysiol, 76, 3524-34.

Merzenich MM, Jenkins WM, Middlebrooks JC (1985) Observations and hypotheses on special organizational features of the central auditory nervous system. In: Dynamic Aspects of Neocortical Function. Edited by E. G. a. W. M. C. G. Edelman. New York: Wiley, pp. 397-423.

Merzenich MM, Knight PL, Roth GL (1973) Cochleotopic organization of primary auditory cortex in the cat. Brain Res, 63, 343-6.

Nelken I, Prut Y, Vaadia E, Abeles M (1994) In search of the best stimulus: an optimization procedure for finding efficient stimuli in the cat auditory cortex. Hear Res, 72, 237-53.

Phillips DP, Cynader MS (1985) Some neural mechanisms in the cat's auditory cortex underlying sensitivity to combined tone and wide-spectrum noise stimuli. Hear Res, 18, 87-102.

Reid RC, Alonso JM (1995) Specificity of monosynaptic connections from thalamus to visual cortex. Nature, 378, 281-4.

Reid RC, Soodak RE, Shapley RM (1991) Directional selectivity and spatiotemporal structure of receptive fields of simple cells in cat striate cortex. J Neurophysiol, 66, 505-29.

Ringach DL, Hawken MJ, Shapley R (1997) Dynamics of orientation tuning in macaque primary visual cortex. Nature, 387, 281-4.

Schreiner CE, Mendelson JR (1990) Functional topography of cat primary auditory cortex: distribution of integrated excitation. J Neurophysiol, 64, 1442-59.

Schreiner CE, Urbas JV (1988) Representation of amplitude in the auditory cortex of the cat. II. Comparison between cortical fields. Hear. Res., 32, 49-64.

Shamma SA, Symmes D (1985) Patterns of inhibition in auditory cortical cells in awake squirrel monkeys. Hear Res, 19, 1-13.

Suga N (1965) Responses of cortical auditory neurones to frequency modulated sounds in echo-locating bats. Nature, 206, 890-1.

Sutter ML, Schreiner CE (1995) Topography of intensity tuning in cat primary auditory cortex: single-neuron versus multiple-neuron recordings. J Neurophysiol, 73, 190-204.

Wang X, Merzenich MM, Beitel R, Schreiner CE (1995) Representation of a species-specific vocalization in the primary auditory cortex of the common marmoset: temporal and spectral characteristics. J Neurophysiol, 74, 2685-706.

Wollberg Z, Newman JD (1972) Auditory cortex of squirrel monkey: response patterns of single cells to species-specific vocalizations. Science, 175, 212-214.

